# Analysis of Drifting Dynamics with Neural Network Hidden Markov Models

| J. Kohlmorgen | K.-R. Müller | K. Pawelzik |
|---|---|---|
| GMD FIRST | GMD FIRST | MPI f. Strömungsforschung |
| Rudower Chaussee 5 | Rudower Chaussee 5 | Bunsenstr. 10 |
| 12489 Berlin, Germany | 12489 Berlin, Germany | 37073 Göttingen, Germany |

## Abstract

We present a method for the analysis of nonstationary time series with multiple operating modes. In particular, it is possible to detect and to model both a switching of the dynamics and a less abrupt, time consuming drift from one mode to another. This is achieved in two steps. First, an unsupervised training method provides prediction experts for the inherent dynamical modes. Then, the trained experts are used in a hidden Markov model that allows to model drifts. An application to physiological wake/sleep data demonstrates that analysis and modeling of real-world time series can be improved when the drift paradigm is taken into account.

## 1 Introduction

Modeling dynamical systems through a measured time series is commonly done by reconstructing the state space with time-delay coordinates [10]. The prediction of the time series can then be accomplished by training neural networks [11]. If, however, a system operates in multiple modes and the dynamics is *drifting* or *switching*, standard approaches like multi-layer perceptrons are likely to fail to represent the underlying input-output relations. Moreover, they do not reveal the dynamical structure of the system. Time series from alternating dynamics of this type can originate from many kinds of systems in physics, biology and engineering.

In [2, 6, 8], we have described a framework for time series from *switching* dynamics, in which an ensemble of neural network predictors specializes on the respective operating modes. We now extend the ability to describe a mode change not only as a switching but – if appropriate – also as a drift from one predictor to another. Our results indicate that physiological signals contain drifting dynamics, which

underlines the potential relevance of our method in time series analysis.

## 2   Detection of Drifts

The detection and analysis of drifts is performed in two steps. First, an unsupervised (hard-)segmentation method is applied. In this approach, an ensemble of competing prediction experts $f_i$, $i = 1, ..., N$, is trained on a given time series. The optimal choice of function approximators $f_i$ depends on the specific application. In general, however, neural networks are a good choice for the prediction of time series [11]. In this paper, we use radial basis function (RBF) networks of the Moody-Darken type [5] as predictors, because they offer a fast and robust learning method.

Under a gaussian assumption, the probability that a particular predictor $i$ would have produced the observed data $y$ is given by

$$p(y \mid i) = Ke^{-\beta(y-f_i)^2}, \tag{1}$$

where $K$ is the normalization term for the gaussian distribution. If we assume that the experts are mutually exclusive and exhaustive, we have $p(y) = \sum_i p(y \mid i)p(i)$. We further assume that the experts are – a priori – equally probable,

$$p(i) = 1/N. \tag{2}$$

In order to train the experts, we want to maximize the likelihood that the ensemble would have generated the time series. This can be done by a gradient method. For the derivative of the log-likelihood $\log L = \log(p(y))$ with respect to the output of an expert, we get

$$\frac{\partial \log L}{\partial f_i} \propto \left[ \frac{e^{-\beta(y-f_i)^2}}{\sum_j e^{-\beta(y-f_j)^2}} \right] (y - f_i). \tag{3}$$

This learning rule can be interpreted as a weighting of the learning rate of each expert by the expert's relative prediction performance. It is a special case of the Mixtures of Experts [1] learning rule, with the gating network being omitted. Note that according to Bayes' rule the term in brackets is the posterior probability that expert $i$ is the correct choice for the given data $y$, i.e. $p(i \mid y)$. Therefore, we can simply write

$$\frac{\partial \log L}{\partial f_i} \propto p(i \mid y)(y - f_i). \tag{4}$$

Furthermore, we imposed a low-pass filter on the prediction errors $\varepsilon_i = (y - f_i)^2$ and used deterministic annealing of $\beta$ in the training process (see [2, 8] for details). We found that these modifications can be essential for a successful segmentation and prediction of time series from switching dynamics.

As a prerequisite of this method, mode changes should occur infrequent, i.e. between two mode changes the dynamics should operate stationary in one mode for a certain number of time steps. Applying this method to a time series yields a (hard) segmentation of the series into different operating modes together with prediction experts for each mode. In case of a drift between two modes, the respective segment tends to be subdivided into several parts, because a single predictor is not able to handle the nonstationarity.

The second step takes the drift into account. A segmentation algorithm is applied that allows to model drifts between two stationary modes by combining the two respective predictors, $f_i$ and $f_j$. The drift is modeled by a weighted superposition

$$f(\vec{x}_t) = a(t) f_i(\vec{x}_t) + (1 - a(t)) f_j(\vec{x}_t), \quad 0 \le a(t) \le 1, \tag{5}$$

where $a(t)$ is a mixing coefficient and $\vec{x}_t = (x_t, x_{t-\tau}, \ldots, x_{t-(m-1)\tau})^T$ is the vector of time-delay coordinates of a (scalar) time series $\{x_t\}$. Furthermore, $m$ is the embedding dimension and $\tau$ is the delay parameter of the embedding. Note that the use of multivariate time series is straightforward.

## 3    A Hidden Markov Model for Drift Segmentation

In the following, we will set up a hidden Markov model (HMM) that allows us to use the Viterbi algorithm for the analysis of drifting dynamics. For a detailed description of HMMs, see [9] and the references therein. An HMM consists of (1) a set $S$ of states, (2) a matrix $A = \{p_{\hat{s},s}\}$ of state transition probabilities, (3) an observation probability distribution $p(y|s)$ for each state $s$, which is a continuous density in our case, and (4) the initial state distribution $\pi = \{\pi_s\}$.

Let us first consider the construction of $S$, the set of states, which is the crucial point of this approach. Consider a set $P$ of 'pure' states (dynamical modes). Each state $s \in P$ represents one of the neural network predictors $f_{k(s)}$ trained in the first step. The predictor of each state performs the predictions autonomously. Next, consider a set $M$ of mixture states, where each state $s \in M$ represents a linear mixture of two nets $f_{i(s)}$ and $f_{j(s)}$. Then, given a state $s \in S, S = P \cup M$, the prediction of the overall system is performed by

$$g_s(\vec{x}_t) = \begin{cases} f_{k(s)}(\vec{x}_t) & ; \text{if } s \in P \\ a(s)f_{i(s)}(\vec{x}_t) + b(s)f_{j(s)}(\vec{x}_t) & ; \text{if } s \in M \end{cases} \tag{6}$$

For each mixture state $s \in M$, the coefficients $a(s)$ and $b(s)$ have to be set together with the respective network indices $i(s)$ and $j(s)$. For computational feasibility, the number of mixture states has to be restricted. Our intention is to allow for drifts between any two network outputs of the previously trained ensemble. We choose $a(s)$ and $b(s)$ such that $0 < a(s) < 1$ and $b(s) = 1 - a(s)$. Moreover, a discrete set of $a(s)$ values has to be defined. For simplicity, we use equally distant steps,

$$a_r = \frac{r}{R+1}, \quad r = 1, \ldots, R. \tag{7}$$

$R$ is the number of intermediate mixture levels. A given resolution $R$ between any two out of $N$ nets yields a total number of mixed states $|M| = R \cdot N \cdot (N-1)/2$. If, for example, the resolution $R = 32$ is used and we assume $N = 8$, then there are $|M| = 896$ mixture states, plus $|P| = N = 8$ pure states.

Next, the transition matrix $A = \{p_{\hat{s},s}\}$ has to be chosen. It determines the transition probability for each pair of states. In principle, this matrix can be found using a training procedure, as e.g. the Baum-Welch method [9]. However, this is hardly feasible in this case, because of the immense size of the matrix. In the above example, the matrix $A$ has $(896 + 8)^2 = 817216$ elements that would have to be estimated. Such an exceeding number of free parameters is prohibitive for any adaptive method. Therefore, we use a *fixed* matrix. In this way, prior knowledge about

the dynamical system can be incorporated. In our applications either switches or smooth drifts between two nets are allowed, in such a way that a (monotonous) drift from one net to another is *a priori* as likely as a switch. All the other transitions are disabled by setting $p_{\tilde{s},s} = 0$. Defining $p(y \mid s)$ and $\pi$ is straightforward. Following eq.(1) and eq.(2), we assume gaussian noise

$$p(y \mid s) = K e^{-\beta(y - g_s)^2},$$ (8)

and equally probable initial states, $\pi_s = |S|^{-1}$.

The Viterbi algorithm [9] can then be applied to the above stated HMM, without any further training of the HMM parameters. It yields the drift segmentation of a given time series, i.e. the most likely state sequence (the sequence of predictors or linear mixtures of two predictors) that could have generated the time series, in our case with the assumption that mode changes occur either as (smooth) drifts or as infrequent switches.

## 4 Drifting Mackey-Glass Dynamics

As an example, consider a high-dimensional chaotic system generated by the Mackey-Glass delay differential equation

$$\frac{dx(t)}{dt} = -0.1x(t) + \frac{0.2x(t - t_d)}{1 + x(t - t_d)^{10}}.$$ (9)

It was originally introduced as a model of blood cell regulation [4]. Two stationary operating modes, A and B, are established by using different delays, $t_d = 17$ and 23, respectively. After operating 100 time steps in mode A (with respect to a subsampling step size $\tau = 6$), the dynamics is drifting to mode B. The drift takes another 100 time steps. It is performed by mixing the equations for $t_d = 17$ and 23 during the integration of eq.(9). The mixture is generated according to eq.(5), using an exponential drift

$$a(t) = \exp\left(\frac{-4\,t}{100}\right), \quad t = 1, \dots, 100.$$ (10)

Then, the system runs stationary in mode B for the following 100 time steps, whereupon it is *switching* back to mode A at $t = 300$, and the loop starts again (Fig.1(a)). The competing experts algorithm is applied to the first 1500 data points of the generated time series, using an ensemble of 6 predictors $f_i(\vec{x}_t)$, $i = 1, ..., 6$. The input to each predictor is a vector $\vec{x}_t$ of time-delay coordinates of the scalar time series $\{x_t\}$. The embedding dimension is $m = 6$ and the delay parameter is $\tau = 1$ on the subsampled data. The RBF predictors consist of 40 basis functions each.

After training, nets 2 and 3 have specialized on mode A, nets 5 and 6 on mode B. This is depicted in the drift segmentation in Fig.1(b). Moreover, the removal of four nets does not increase the root mean squared error (RMSE) of the prediction significantly (Fig.1(c)), which correctly indicates that two predictors completely describe the dynamical system. The sequence of nets to be removed is obtained by repeatedly computing the RMSE of all $n$ subsets with $n - 1$ nets each, and then selecting the subset with the lowest RMSE of the respective drift segmentation. The segmentation of the remaining nets, 2 and 5, nicely reproduces the evolution of the dynamics, as seen in Fig.1(d).

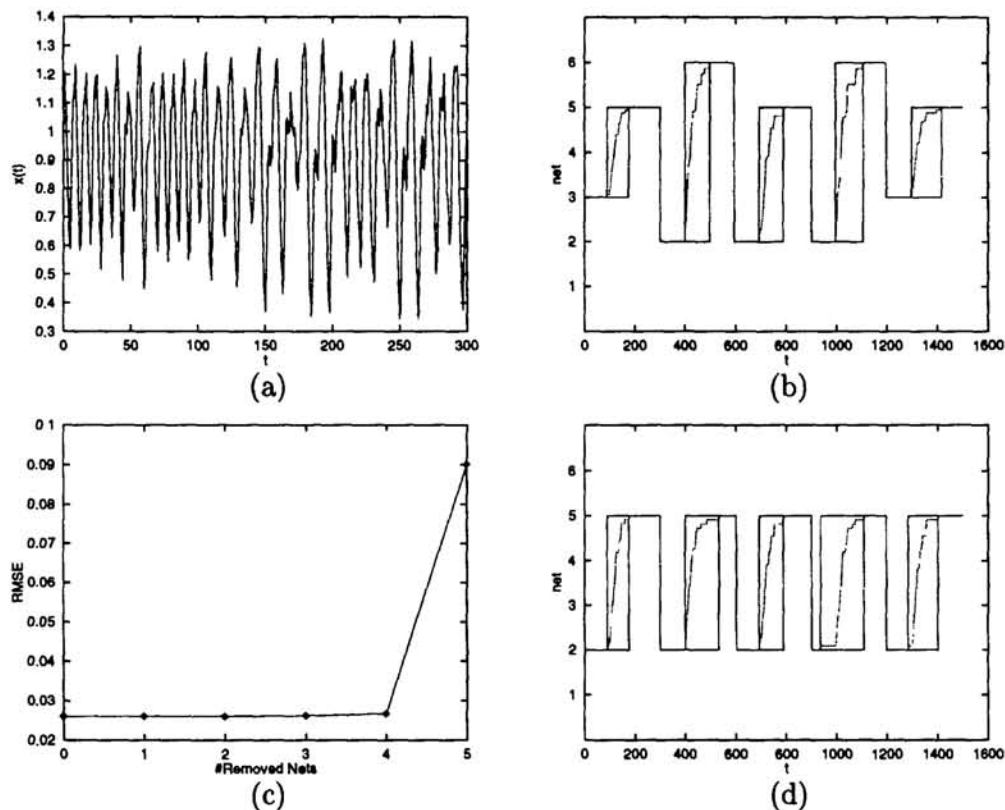

Figure 1: *(a) One 'loop' of the drifting Mackey-Glass time series (see text). (b) The resulting drift segmentation invokes four nets. The dotted line indicates the evolution of the mixing coefficient a(t) of the respective nets. For example, between t = 100 and 200 it denotes a drift from net 3 to net 5, which appears to be exponential. (c) Increase of the prediction error when predictors are successively removed. (d) The two remaining predictors model the dynamics of the time series properly.*

## 5   Wake/Sleep EEG

In [7], we analyzed physiological data recorded from the wake/sleep transition of a human. The objective was to provide an unsupervised method to detect the sleep onset and to give a detailed approximation of the signal dynamics with a high time resolution, ultimately to be used in diagnosis and treatment of sleep disorders. The application of the drift segmentation algorithm now yields a more detailed modeling of the dynamical system.

As an example, Fig. 2 shows a comparison of the drift segmentation ($R = 32$) with a manual segmentation by a medical expert. The experimental data was measured during an afternoon nap of a healthy human. The computer-based analysis is performed on a single-channel EEG recording (occipital-1), whereas the manual segmentation was worked out using several physiological recordings (EEG, EOG, ECG, heart rate, blood pressure, respiration).

The two-step drift segmentation method was applied using 8 RBF networks. However, as shown in Fig. 2, three nets (4, 6, and 8) are finally found by the Viterbi algorithm to be sufficient to represent the most likely state sequence. Before the sleep onset, at $t \approx 3500$ (350s) in the manual analysis, a mixture of two wake-state

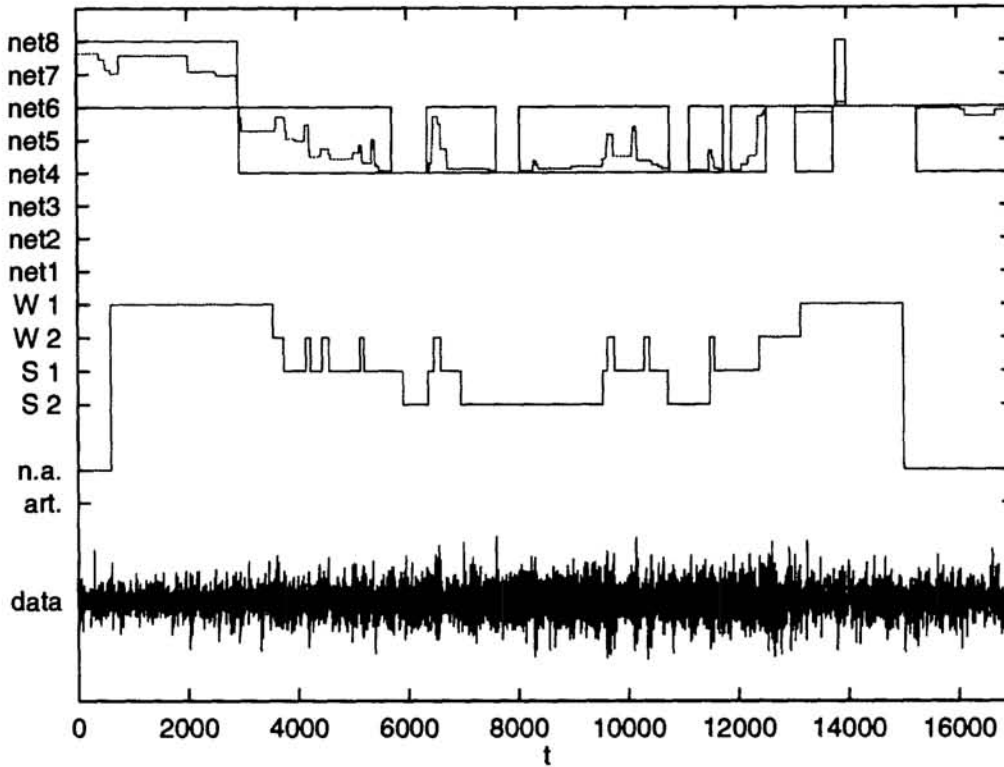

Figure 2: *Comparison of the drift segmentation obtained by the algorithm (upper plot), and a manual segmentation by a medical expert (middle). Only a single-channel EEG recording (occipital-1, time resolution 0.1s) of an afternoon nap is given for the algorithmic approach, while the manual segmentation is based on all available measurements. In the manual analysis, W1 and W2 indicate two wake-states (eyes open/closed), and S1 and S2 indicate sleep stage I and II, respectively. (n.a.: no assessment, art.: artifacts)*

nets, 6 and 8, performs the best reconstruction of the EEG dynamics. Then, at $t = 3000$ (300s), there starts a drift to net 4, which apparently represents the dynamics of sleep stage II (S2). Interestingly, sleep stage I (S1) is not represented by a separate net but by a linear mixture of net 4 and net 6, with much more weight on net 4. Thus, the process of falling asleep is represented as a drift from the state of being awake directly to sleep stage II.

During sleep there are several wake-up spikes indicated in the manual segmentation. At least the last four are also clearly indicated in the drift segmentation, as drifts back to net 6. Furthermore, the detection of the final arousal after $t = 12000$ (1200s) is in good accordance with the manual segmentation: there is a fast drift back to net 6 at that point.

Considering the fact that our method is based only on the recording of a single EEG channel and does not use any medical expert knowledge, the drift algorithm is in remarkable accordance with the assessment of the medical expert. Moreover, it resolves the dynamical structure of the signal to more detail. For a more comprehensive analysis of wake/sleep data, we refer to our forthcoming publication [3].

# 6   Summary and Discussion

We presented a method for the unsupervised segmentation and identification of nonstationary drifting dynamics. It applies to time series where the dynamics is drifting or switching between different operating modes. An application to physiological wake/sleep data (EEG) demonstrates that drift can be found in natural systems. It is therefore important to consider this aspect of data description.

In the case of wake/sleep data, where the physiological state transitions are far from being understood, we can extract the shape of the dynamical drift from wake to sleep in an unsupervised manner. By applying this new data analysis method, we hope to gain more insights into the underlying physiological processes. Our future work is therefore dedicated to a comprehensive analysis of large sets of physiological wake/sleep recordings. We expect, however, that our method will be also applicable in many other fields.

**Acknowledgements:** We acknowledge support of the DFG (grant Ja379/51) and we would like to thank J. Rittweger for the EEG data and for fruitful discussions.

# References

[1] Jacobs, R.A., Jordan, M.A., Nowlan, S.J., Hinton, G.E. (1991). Adaptive Mixtures of Local Experts, *Neural Computation* **3**, 79–87.

[2] Kohlmorgen, J., Müller, K.-R., Pawelzik, K. (1995). Improving short-term prediction with competing experts. ICANN'95, EC2 & Cie, Paris, 2:215–220.

[3] Kohlmorgen, J., Müller, K.-R., Rittweger, J., Pawelzik, K., in preparation.

[4] Mackey, M., Glass, L. (1977). Oscillation and Chaos in a Physiological Control System, Science **197**, 287.

[5] Moody, J., Darken, C. (1989). Fast Learning in Networks of Locally-Tuned Processing Units. *Neural Computation* **1**, 281–294.

[6] Müller, K.-R., Kohlmorgen, J., Pawelzik, K. (1995). Analysis of Switching Dynamics with Competing Neural Networks, *IEICE Trans. on Fundamentals of Electronics, Communications and Computer Sc.*, E78-A, No.10, 1306–1315.

[7] Müller, K.-R., Kohlmorgen, J., Rittweger, J., Pawelzik, K. (1995). Analysing Physiological Data from the Wake-Sleep State Transition with Competing Predictors, NOLTA'95: Symposium on Nonlinear Theory and its Appl., 223–226.

[8] Pawelzik, K., Kohlmorgen, J., Müller, K.-R. (1996). Annealed Competition of Experts for a Segmentation and Classification of Switching Dynamics, *Neural Computation*, **8:2**, 342–358.

[9] Rabiner, L.R. (1988). A Tutorial on Hidden Markov Models and Selected Applications in Speech Recognition. In *Readings in Speech Recognition*, ed. A. Waibel, K. Lee, 267–296. San Mateo: Morgan Kaufmann, 1990.

[10] Takens, F. (1981). Detecting Strange Attractors in Turbulence. In: Rand, D., Young, L.-S., (Eds.), *Dynamical Systems and Turbulence*, Springer Lecture Notes in Mathematics, **898**, 366.

[11] Weigend, A.S., Gershenfeld, N.A. (Eds.) (1994). *Time Series Prediction: Forecasting the Future and Understanding the Past*, Addison-Wesley.
